# Supervised Learning with Growing Cell Structures

**Bernd Fritzke**
Institut für Neuroinformatik
Ruhr-Universität Bochum
Germany

## Abstract

We present a new incremental radial basis function network suitable for classification and regression problems. Center positions are continuously updated through soft competitive learning. The width of the radial basis functions is derived from the distance to topological neighbors. During the training the observed error is accumulated locally and used to determine where to insert the next unit. This leads (in case of classification problems) to the placement of units near class borders rather than near frequency peaks as is done by most existing methods. The resulting networks need few training epochs and seem to generalize very well. This is demonstrated by examples.

## 1 INTRODUCTION

Feed-forward networks of localized (e.g., Gaussian) units are an interesting alternative to the more frequently used networks of global (e.g., sigmoidal) units. It has been shown that with localized units one hidden layer suffices in principle to approximate any continuous function, whereas with sigmoidal units two layers are necessary.

In the following we are considering radial basis function networks similar to those proposed by Moody & Darken (1989) or Poggio & Girosi (1990). Such networks consist of one layer $L$ of Gaussian units. Each unit $c \in L$ has an associated vector $w_c \in R^n$ indicating the position of the Gaussian in input vector space and a standard

deviation $\sigma_c$. For a given input datum $\xi \in R^n$ the activation of unit $c$ is described by

$$D_c(\xi) = \exp\left(-\frac{\|\xi - w_c\|^2}{\sigma_c^2}\right).$$ (1)

On top of the layer $L$ of Gaussian units there are $m$ *single layer perceptrons*. Thereby, $m$ is the output dimensionality of the problem which is given by a number of input/output pairs[1] $(\xi, \zeta) \in (R^n \times R^m)$. Each of the single layer perceptrons computes a weighted sum of the activations in $L$:

$$O_i(\xi) = \sum_{j \in L} w_{ij} D_j(\xi) \qquad i \in \{1, \ldots, m\}$$ (2)

With $w_{ij}$ we denote the weighted connection from local unit $j$ to output unit $i$. Training of a single layer perceptron to minimize square error is a very well understood problem which can be solved incrementally by the delta rule or directly by linear algebra techniques (Moore-Penrose inverse). Therefore, the only (but severe) difficulty when using radial basis function networks is choosing the number of local units and their respective parameters, namely center position $w$ and width $\sigma$.

One extreme approach is to use one unit per data points and to position the units directly at the data points. If one chooses the width of the Gaussians sufficiently small it is possible to construct a network which correctly classifies the training data, no matter how complicated the task is (Fritzke, 1994). However, the network size is very large and might even be infinite in the case of a continuous stream of non-repeating stochastic input data. Moreover, such a network can be expected to generalize poorly.

Moody & Darken (1989), in contrast, propose to use a fixed number of local units (which is usually considerably smaller than the total number of data points). These units are first distributed by an unsupervised clustering method (e.g., k-means). Thereafter, the weights to the output units are determined by gradient descent. Although good results are reported for this method it is rather easy to come up with examples where it would not perform well: k-means positions the units based on the density of the training data, specifically *near density peaks*. However, to approximate the optimal Bayesian a posteriori classifier it would be better to position units *near class borders*. Class borders, however, often lie in regions with a particularly low data density. Therefore, all methods based on k-means-like unsupervised placement of the Gaussians are in danger to perform poorly with a fixed number of units or – similarly undesirable – to need a huge number of units to achieve decent performance.

From this one can conclude that – in the case of radial basis function networks – it is essential to use the class labels not only for the training of the connection weights but also for the placement of the local units. Doing this forms the core of the method proposed below.

# 2   SUPERVISED GROWING CELL STRUCTURES

In the following we present an incremental radial basis function network which is able to simultaneously determine a suitable number of local units, their center positions and widths as well as the connection weights to the output units. The basic idea is a very simple one:

0. Start with a very small radial basis function network.

1. Train the current network with some I/O-pairs from the training data.

2. Use the observed accumulated error to determine where in input vector space to insert new units.

3. If network does not perform well enough goto 1.

One should note that during the training phase (Step 1.) error is accumulated over several data items and this accumulated error is used to determine where to insert new units (Step 2.). This is different from the approach of Platt (1991) where insertions are based on single poorly mapped patterns. In both cases, however, the goal is to position new units in regions where the current network does not perform well rather than in regions where many data items stem from.

In our model the center positions of new units are *interpolated* from the positions of existing units. Specifically, after some adaptation steps we determine the unit $q$ which has accumulated the maximum error and insert a new unit in between $q$ and one of its neighbors in input vector space. The interpolation procedure makes it necessary to allow the center positions of existing units to change. Otherwise, all new units would be restricted to the convex hull of the centers of the initial network.

We do not necessarily insert a new unit in between $q$ and its nearest neighbor. Rather we like to choose one of the units with adjacent *Voronoi regions*[2]. In the two-dimensional case these are the direct neighbors of $q$ in the *Delaunay triangulation* (Delaunay-neighbors) induced by all center positions. In higher-dimensional spaces there exists an equivalent based on hypertetrahedrons which, however, is very hard to compute. For this reason, we arrange our units in a certain topological structure (see below) which has the property that if two units are direct neighbors in that structure they are *mostly* Delaunay-neighbors. By this we get with very little computational effort an approximate subset of the Delaunay-neighbors which seems to be sufficient for practical purposes.

## 2.1   NETWORK STRUCTURE

The structure of our network is very similar to standard radial basis function networks. The only difference is that we arrange the local units in a $k$-dimensional topological structure consisting of connected simplices[3] (lines for $k = 1$, triangles

for $k = 2$, tetrahedrons for $k = 3$ and hypertetrahedrons for larger $k$). This arrangement is done to facilitate the interpolation and adaptation steps described below. The initial network consists of *one* $k$-dimensional simplex ($k + 1$ local units fully connected with each other). The neighborhood connections are not weighted and do not directly influence the behavior of the network. They are, however, used to determine the width of the Gaussian functions associated with the units. Let for each Gaussian unit $c$ denote $N_c$ the set of direct topological neighbors in the topological structure. Then the width of $c$ is defined as

$$\sigma_c = (1/|N_c|) \sum_{d \in N_c} \|w_c - w_d\|_2 \tag{3}$$

which is the mean distance to the topological neighbors. If topological neighbors have similar center positions (which will be ensured by the way adaptation and insertion is done) then this leads to a covering of the input vector space with partially overlapping Gaussian functions.

## 2.2 ADAPTATION

It was mentioned above that several adaptation steps are done before a new unit is inserted. One single adaptation step is done as follows (see fig. 1):

- Chose an I/O-pair $(\xi, \zeta), \xi \in R^n, \zeta \in R^m)$ from the training data.
- determine the unit $s$ closest to $\xi$ (the so-called best-matching unit).
- Move the centers of $s$ and its direct topological neighbors towards $\xi$.

$$\Delta w_s = \varepsilon_b(\xi - w_s), \qquad \Delta w_c = \varepsilon_n(\xi - w_c) \quad \text{for all } c \in N_s$$

$\varepsilon_b$ and $\varepsilon_n$ are small constants with $\varepsilon_b \gg \varepsilon_n$.

- Compute for each local unit $c \in L$ the activation $D_c(\xi)$    (see eqn. 1)
- Compute for each output unit $i$ the activation $O_i$    (see eqn. 2)
- Compute the square error by

$$\text{SE} = \sum_{i=1}^{m}(\zeta_i - O_i)^2$$

- Accumulate error at best-matching unit $s$:

$$\Delta\text{err}_s = \text{SE}$$

- Make Delta-rule step for the weights ($\alpha$ denotes the learning rate):

$$\Delta w_{ij} = \alpha D_j(\zeta_i - O_i) \qquad i \in \{1, \ldots, m\}, j \in L$$

Since together with the best-matching unit always its direct topological neighbors are adapted, neighboring units tend to have similar center positions. This property can be used to determine suitable center positions for new units as will be demonstrated in the following.

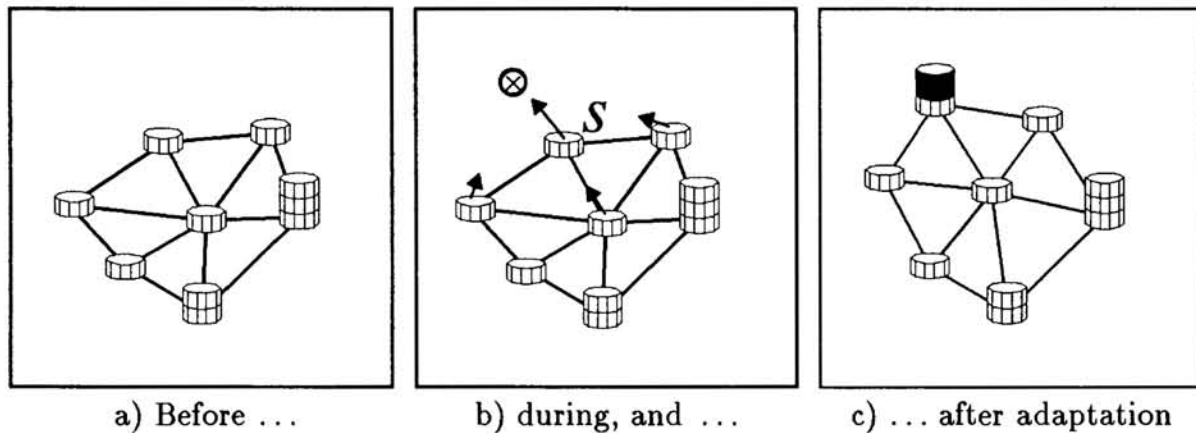

a) Before ...            b) during, and ...            c) ... after adaptation

Figure 1: One adaptation step. The center positions of the current network are shown and the change caused by a single input signal. The observed error SE for this pattern is added to the local error variable of the best-matching unit.

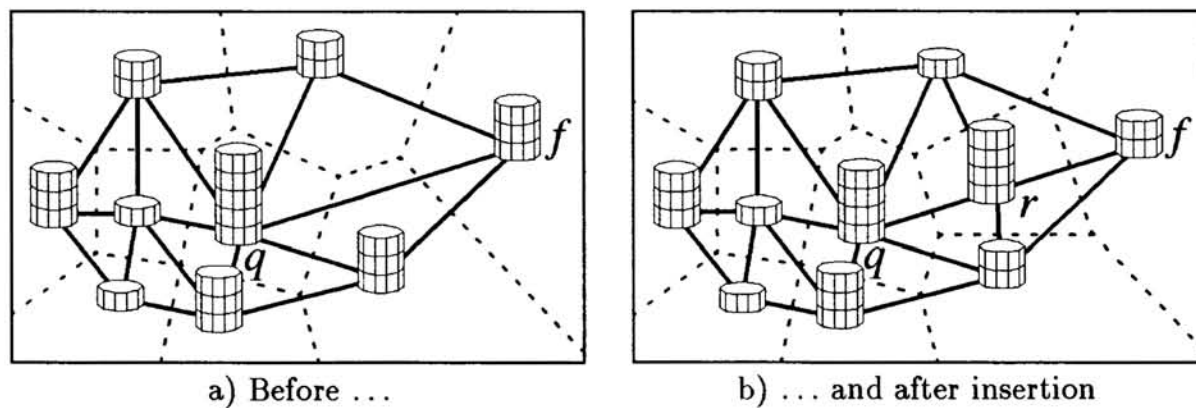

a) Before ...                          b) ... and after insertion

Figure 2: Insertion of a new unit. The dotted lines indicate the Voronoi fields. The unit $q$ has accumulated the most error and, therefore, a new unit is inserted between $q$ and one of its direct neighbors.

## 2.3   INSERTION OF NEW UNITS

After a constant number $\lambda$ of adaptation steps a new unit is inserted. For this purpose the unit $q$ with maximum accumulated error is determined. Obviously, $q$ lies in a region of the input vector space where many misclassifications occur. One possible reason for this is that the gradient descent procedure is unable to find suitable weights for the current network. This again might be caused by the coarse resolution at this region of the input vector space: if data items from different classes are covered by the same local unit and activate this unit to about the same degree then it might be the case that their vectors of local unit activations are nearly identical which makes it hard for the following single layer perceptrons to distinguish among them. Moreover, even if the activation vectors are sufficiently different they still might be not linearly separable.

---

accurate) approximation of the Delaunay triangulation which is based on the "Neural-Gas" method proposed by Martinetz & Schulten (1991).

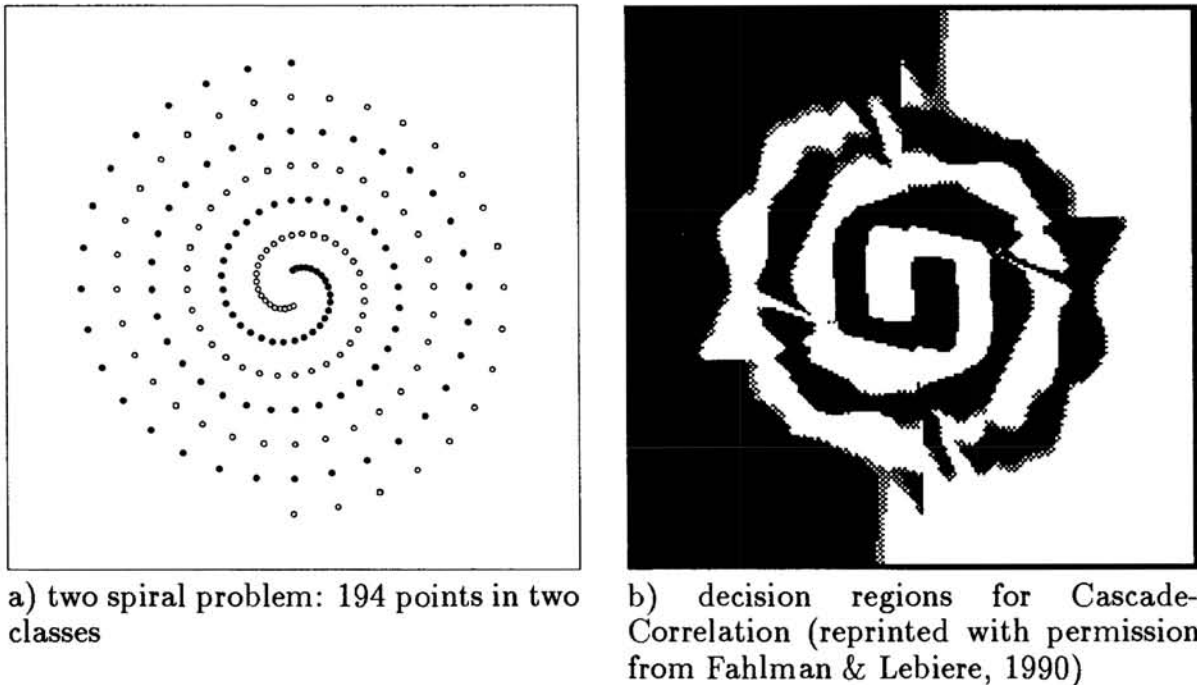

a) two spiral problem: 194 points in two classes

b) decision regions for Cascade-Correlation (reprinted with permission from Fahlman & Lebiere, 1990)

Figure 3: Two spiral problem and learning results of a constructive network.

The insertion of a new local unit near $q$ is likely to improve the situation: This unit will probably be activated to a different degree by the data items in this region and will, therefore, make the problem easier for the single layer perceptrons.

What exactly are we doing? We choose one of the direct topological neighbors of $q$, say a unit $f$ (see also fig. 2). Currently this is the neighbor with the maximum accumulated error. Other choices, however, have shown good results as well, e.g., the neighbor with the most distant center position or even a randomly picked neighbor. We insert a new unit $r$ in between $q$ and $f$ and initialize its center by

$$w_r = (w_q + w_f)/2 \tag{4}$$

We connect the new unit with $q$ and $f$ and with all common neighbors of $q$ and $f$. The original connection between $q$ and $f$ is removed. By this we get a structure of $k$-dimensional simplices again. The new unit gets weights to the output units which are interpolated from the weights of its neighbors. The same is done for the initial error variable which is linearly interpolated from the variables of the neighbors of $r$. After the interpolation all the weights of $r$ and its neighbors and the error variables of these units are multiplied by a factor $|N_r|/(|N_r| + 1)|$. This is done to disturb the output of the network as less as possible[4]. However, the by far most important

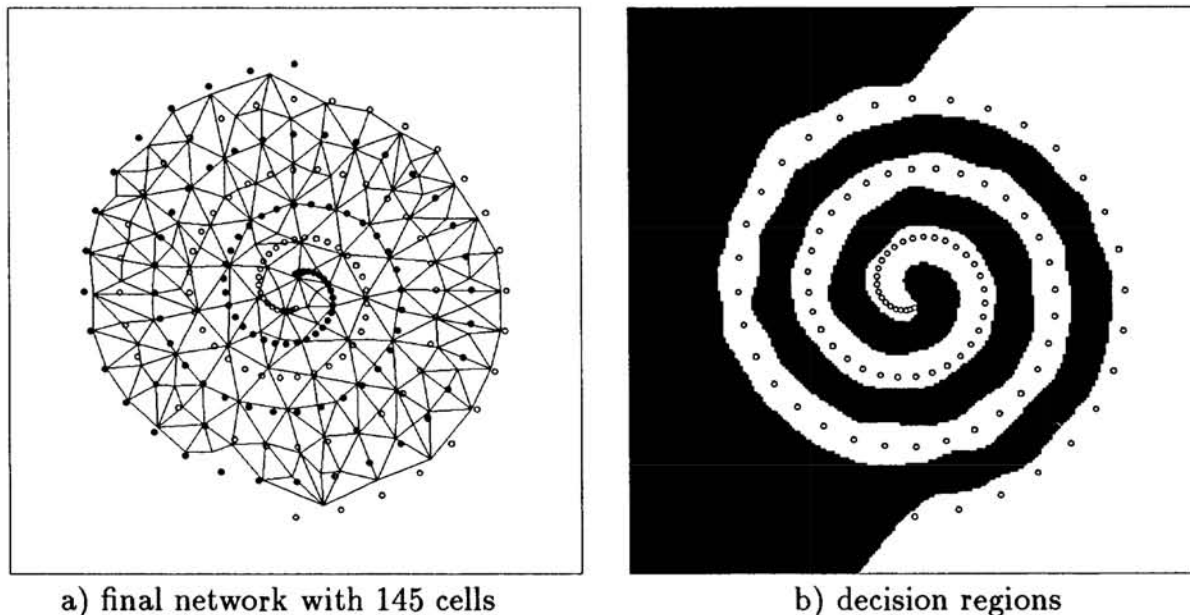

| a) final network with 145 cells | b) decision regions |

Figure 4: Performance of the Growing Cell Structures on the two spiral benchmark.

decision seems to be to insert the new unit near the unit with maximum error. The weights and the error variables adjust quickly after some learning steps.

## 2.4    SIMULATION RESULTS

Simulations with the two spiral problem (fig. 3a) have been performed. This classification benchmark has been widely used before so that results for comparison are readily available.Figure 3b) shows the result of another constructive algorithm. The data consist of 194 points arranged on two interlaced spirals in the plane. Each spiral corresponds to one class. Due to the high nonlinearity of the task it is particular difficult for networks consisting of global units (e.g., multi-layer perceptrons). However, the varying density of data points (which is higher in the center of the spirals) makes it also a challenge for networks of local units.

As for most learning problems the interesting aspect is not learning the training examples but rather the performance on new data which is often denoted as generalization. Baum & Lang (1991) defined a test set of 576 points for this problem consisting of three equidistant test points between each pair of adjacent same-class training points. They reported for their best network 29 errors on the test set in the mean.

In figure 4 a typical network generated by our method can be seen as well as the corresponding decision regions. No errors on the test set of Baum and Lang are made. Table 1 shows the necessary training cycles for several algorithms. The new growing network uses far less cycles than the other networks.

Other experiments have been performed with a vowel recognition problem (Fritzke, 1993). In all simulations we obtained significantly better generalization results than Robinson (1989) who in his thesis investigated the performance of several connectionist and conventional algorithms on the same problem. The necessary

Table 1: Training epochs necessary for the two spiral problem

| network model | epochs | test error | reported in |
| --- | --- | --- | --- |
| Backpropagation | 20000 | yes | Lang & Witbrock (1989) |
| Cross Entropy BP | 10000 | yes | Lang & Witbrock (1989) |
| Cascade-Correlation | 1700 | yes | Fahlman & Lebiere (1990) |
| Growing Cell Structures | 180 | no | Fritzke (1993) |

number of training cycles for our method was lower by a factor of about 37 than the numbers reported by Robinson (1993, personal communication).

## Footnotes

[1]Throughout this article we assume a classification problem and use the corresponding terminology. However, the described method is suitable for regression problems as well.

[2]The Voronoi region of a unit $c$ denotes the part of the input vector space which consists of points for which $c$ is the nearest unit.

[3]A historical reason for this specific approach is the fact that the model was developed from an unsupervised network (see Fritzke, 1993) where the $k$-dimensional neighborhood was needed to reduce dimensionality. We currently investigate an alternative (and more

[4]The redistribution of the error variable is again a relict from the unsupervised version (Fritzke, 1993). There we count signals rather than accumulate error. An elaborate scheme for redistributing the signal counters is necessary to get good local estimates of the probability density. For the supervised version this redistribution is harder to justify since the insertion of a new unit in general makes previous error information void. However, even though there is still some room for simplification, the described scheme does work very well already in its present form.

## REFERENCES

Baum, E. B. & K. E. Lang [1991], "Constructing hidden units using examples and queries," in *Advances in Neural Information Processing Systems 3*, R.P. Lippmann, J.E. Moody & D.S. Touretzky, eds., Morgan Kaufmann Publishers, San Mateo, 904–910.

Fahlman, S. E. & C. Lebiere [1990], "The Cascade-Correlation Learning Architecture," in *Advances in Neural Information Processing Systems 2*, D.S. Touretzky, ed., Morgan Kaufmann Publishers, San Mateo, 524–532.

Fritzke, B. [1993], "Growing Cell Structures – a self-organizing network for unsupervised and supervised learning," International Computer Science Institute, TR-93-026, Berkeley.

Fritzke, B. [1994], "Making hard problems linearly separable – incremental radial basis function approaches," (*submitted to ICANN'94: International Conference on Artificial Neural Networks*), Sorrento, Italy.

Lang, K. J. & M. J. Witbrock [1989], "Learning to tell two spirals apart," in *Proceedings of the 1988 Connectionist Models Summer School*, D. Touretzky, G. Hinton & T. Sejnowski, eds., Morgan Kaufmann, San Mateo, 52–59.

Martinetz, T. M. & K. J. Schulten [1991], "A "neural-gas" network learns topologies," in *Artificial Neural Networks*, T. Kohonen, K. Mäkisara, O. Simula & J. Kangas, eds., North-Holland, Amsterdam, 397–402.

Moody, J. & C. Darken [1989], "Learning with Localized Receptive Fields," in *Proceedings of the 1988 Connectionist Models Summer School*, D. Touretzky, G. Hinton & T. Sejnowski, eds., Morgan Kaufmann, San Mateo, 133–143.

Platt, J. C. [1991], "A Resource-Allocating Network for Function Interpolation," *Neural Computation* 3, 213–225.

Poggio, T. & F. Girosi [1990], "Regularization Algorithms for Learning That Are Equivalent to Multilayer Networks," *Science* 247, 978–982.

Robinson, A. J. [1989], "Dynamic Error Propagation Networks," Cambridge University, PhD Thesis, Cambridge.